# Efficient Online Inference for Bayesian Nonparametric Relational Models

**Dae Il Kim**[1], **Prem Gopalan**[2], **David M. Blei**[2], and **Erik B. Sudderth**[1]

[1]Department of Computer Science, Brown University, {`daeil,sudderth`}`@cs.brown.edu`
[2]Department of Computer Science, Princeton University, {`pgopalan,blei`}`@cs.princeton.edu`

## Abstract

Stochastic block models characterize observed network relationships via latent community memberships. In large social networks, we expect entities to participate in multiple communities, and the number of communities to grow with the network size. We introduce a new model for these phenomena, the hierarchical Dirichlet process relational model, which allows nodes to have mixed membership in an unbounded set of communities. To allow scalable learning, we derive an online stochastic variational inference algorithm. Focusing on assortative models of undirected networks, we also propose an efficient structured mean field variational bound, and online methods for automatically pruning unused communities. Compared to state-of-the-art online learning methods for parametric relational models, we show significantly improved perplexity and link prediction accuracy for sparse networks with tens of thousands of nodes. We also showcase an analysis of Little-Sis, a large network of who-knows-who at the heights of business and government.

## 1 Introduction

A wide range of statistical models have been proposed for the discovery of hidden communities within observed networks. The simplest *stochastic block models* [20] create communities by clustering nodes, aiming to identify demographic similarities in social networks, or proteins with related functional interactions. The *mixed-membership stochastic blockmodel* (MMSB) [1] allows nodes to be members of multiple communities; this generalization substantially improves predictive accuracy in real-world networks. These models are practically limited by the need to externally specify the number of latent communities. We propose a novel *hierarchical Dirichlet process relational* (HDPR) model, which allows mixed membership in an unbounded collection of latent communities. By adapting the HDP [18], we allow data-driven inference of the number of communities underlying a given network, and growth in the community structure as additional nodes are observed.

The *infinite relational model* (IRM) [10] previously adapted the Dirichlet process to define a non-parametric relational model, but restrictively associates each node with only one community. The more flexible *nonparametric latent feature model* (NLFM) [14] uses an Indian buffet process (IBP) [7] to associate nodes with a subset of latent communities. The *infinite multiple membership relational model* (IMRM) [15] also uses an IBP to allow multiple memberships, but uses a non-conjugate observation model to allow more scalable inference for sparse networks. The *nonparametric metadata dependent relational* (NMDR) model [11] employs a logistic stick-breaking prior on the node-specific community frequencies, and thereby models relationships between communities and metadata. All of these previous nonparametric relational models employed MCMC learning algorithms. In contrast, the conditionally conjugate structure of our HDPR model allows us to easily develop a stochastic variational inference algorithm [17, 2, 9]. Its online structure, which incrementally updates global community parameters based on random subsets of the full graph, is highly scalable; our experiments consider social networks with tens of thousands of nodes.

While the HDPR is more broadly applicable, our focus in this paper is on *assortative* models for undirected networks, which assume that the probability of linking distinct communities is small. This modeling choice is appropriate for the clustered relationships found in friendship and collaboration networks. Our work builds on stochastic variational inference methods developed for the assortative MMSB (aMMSB) [6], but makes three key technical innovations. First, adapting work on HDP topic models [19], we develop a nested family of variational bounds which assign positive probability to dynamically varying subsets of the unbounded collection of global communities. Second, we use these nested bounds to dynamically prune unused communities, improving computational speed, predictive accuracy, and model interpretability. Finally, we derive a structured mean field variational bound which models dependence among the pair of community assignments associated with each edge. Crucially, this avoids the expensive and inaccurate local optimizations required by naive mean field approximations [1, 6], while maintaining computation and storage requirements that scale linearly (rather than quadratically) with the number of hypothesized communities.

In this paper, we use our assortative HDPR (aHDPR) model to recover latent communities in social networks previously examined with the aMMSB [6], and demonstrate substantially improved perplexity scores and link prediction accuracy. We also use our learned community structure to visualize business and governmental relationships extracted from the LittleSis database [13].

## 2 Assortative Hierarchical Dirichlet Process Relational Models

We introduce the assortative HDP relational (aHDPR) model, a nonparametric generalization of the aMMSB for discovering shared memberships in an unbounded collection of latent communities. We focus on undirected binary graphs with $N$ nodes and $E = N(N-1)/2$ possible edges, and let $y_{ij} = y_{ji} = 1$ if there is an edge between nodes $i$ and $j$. For some experiments, we assume the $y_{ij}$ variables are only partially observed to compare the predictive performance of different models.

As summarized in the graphical models of Fig. 1, we begin by defining a global Dirichlet process to capture the parameters associated with each community. Letting $\beta_k$ denote the expected frequency of community $k$, and $\gamma > 0$ the concentration, we define a stick-breaking representation of $\beta$:

$$\beta_k = v_k \prod_{\ell=1}^{k-1} (1 - v_\ell), \qquad v_k \sim \text{Beta}(1, \gamma), \qquad k = 1, 2, \dots \tag{1}$$

Adapting a two-layer hierarchical DP [18], the mixed community memberships for each node $i$ are then drawn from DP with base measure $\beta$, $\pi_i \sim \text{DP}(\alpha\beta)$. Here, $\mathbb{E}[\pi_i \mid \alpha, \beta] = \beta$, and small precisions $\alpha$ encourage nodes to place most of their mass on a sparse subset of communities.

To generate a possible edge $y_{ij}$ between nodes $i$ and $j$, we first sample a pair of indicator variables from their corresponding community membership distributions, $s_{ij} \sim \text{Cat}(\pi_i)$, $r_{ij} \sim \text{Cat}(\pi_j)$. We then determine edge presence as follows:

$$p(y_{ij} = 1 \mid s_{ij} = r_{ij} = k) = w_k, \qquad p(y_{ij} = 1 \mid s_{ij} \neq r_{ij}) = \epsilon. \tag{2}$$

For our assortative aHDPR model, each community has its own self-connection probability $w_k \sim \text{Beta}(\tau_a, \tau_b)$. To capture the sparsity of real networks, we fix a very small probability of between-community connection, $\epsilon = 10^{-30}$. Our HDPR model could easily be generalized to more flexible likelihoods in which each pair of communities $k, \ell$ have their own interaction probability [1], but motivated by work on the aMMSB [6], we do not pursue this generalization here.

## 3 Scalable Variational Inference

Previous applications of the MMSB associate a pair of community assignments, $s_{ij}$ and $r_{ij}$, with each potential edge $y_{ij}$. In assortative models these variables are strongly dependent, since present edges only have non-negligible probability for consistent community assignments. To improve accuracy and reduce local optima, we thus develop a structured variational method based on joint configurations of these assignment pairs, which we denote by $e_{ij} = (s_{ij}, r_{ij})$. See Figure 1.

Given this alternative representation, we aim to approximate the joint distribution of the observed edges $y$, local community assignments $e$, and global community parameters $\pi, w, \beta$ given fixed

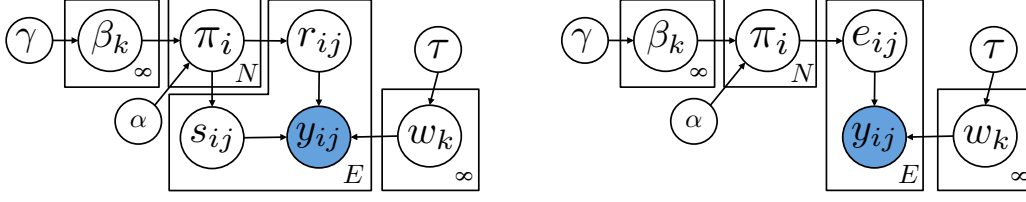

Figure 1: Alternative graphical representations of the aHDPR model, in which each of $N$ nodes has mixed membership $\pi_i$ in an unbounded set of latent communities, $w_k$ are the community self-connection probabilities, and $y_{ij}$ indicates whether an edge is observed between nodes $i$ and $j$. *Left:* Conventional representation, in which source $s_{ij}$ and receiver $r_{ij}$ community assignments are independently sampled. *Right:* Blocked representation in which $e_{ij} = (s_{ij}, r_{ij})$ denotes the pair of community assignments underlying $y_{ij}$.

hyperparameters $\tau, \alpha, \gamma$. Mean field variational methods minimize the KL divergence between a family of approximating distributions $q(e, \pi, w, \beta)$ and the true posterior, or equivalently maximize the following *evidence lower bound* (ELBO) on the marginal likelihood of the observed edges $y$:

$$\mathcal{L}(q) \triangleq \mathbb{E}_q[\log p(y, e, \pi, w, \beta \mid \tau, \alpha, \gamma)] - \mathbb{E}_q[\log q(e, \pi, w, \beta)]. \tag{3}$$

For the nonparametric aHDPR model, the number of latent community parameters $w_k, \beta_k$, and the dimensions of the community membership distributions $\pi_i$, are both infinite. Care must thus be taken to define a tractably factorized, and finitely parameterized, variational bound.

## 3.1  Variational Bounds via Nested Truncations

We begin by defining categorical edge assignment distributions $q(e_{ij} \mid \phi_{ij}) = \text{Cat}(e_{ij} \mid \phi_{ij})$, where $\phi_{ijk\ell} = q(e_{ij} = (k, \ell)) = q(s_{ij} = k, r_{ij} = \ell)$. For some *truncation level* $K$, which will be dynamically varied by our inference algorithms, we constrain $\phi_{ijk\ell} = 0$ if $k > K$ or $\ell > K$. Given this restriction, all observed interactions are explained by one of the first (and under the stick-breaking prior, most probable) $K$ communities. The resulting variational distribution has $K^2$ parameters. This truncation approach extends prior work for HDP topic models [19, 5].

For the global community parameters, we define an *untruncated* factorized variational distribution:

$$q(\beta, w \mid v^*, \lambda) = \prod_{k=1}^{\infty} \delta_{v_k^*}(v_k)\text{Beta}(w_k \mid \lambda_{ka}, \lambda_{kb}), \qquad \beta_k(v^*) = v_k^* \prod_{\ell=1}^{k-1}(1 - v_\ell^*). \tag{4}$$

Our later derivations show that for communities $k > K$ above the truncation level, the optimal variational parameters equal the prior: $\lambda_{ka} = \tau_a, \lambda_{kb} = \tau_b$. These distributions thus need not be explicitly represented. Similarly, the objective only depends on $v_k^*$ for $k \leq K$, defining $K + 1$ probabilities: the frequencies of the first $K$ communities, and the aggregate frequency of all others. Matched to this, we associate a $(K + 1)$-dimensional community membership distribution $\pi_i$ to each node, where the final component contains the sum of all mass not assigned to the first $K$. Exploiting the fact that the Dirichlet process induces a Dirichlet distribution on any finite partition, we let $q(\pi_i \mid \theta_i) = \text{Dir}(\pi_i \mid \theta_i), \theta_i \in \mathbb{R}^{K+1}$. The overall variational objective is then

$$\mathcal{L}(q) = \sum_k \mathbb{E}_q[\log p(w_k \mid \tau_a, \tau_b)] - \mathbb{E}_q[\log q(w_k \mid \lambda_{ka}, \lambda_{kb})] + \mathbb{E}_q[\log p(v_k^* \mid \gamma)] \tag{5}$$
$$+ \sum_i \mathbb{E}_q[\log p(\pi_i \mid \alpha, \beta(v^*))] - \mathbb{E}_q[\log q(\pi_i \mid \theta_i)]$$
$$+ \sum_{ij} \mathbb{E}_q[\log p(y_{ij}|w, e_{ij})] + \mathbb{E}_q[\log p(e_{ij}|\pi_i, \pi_j)] - \mathbb{E}_q[\log q(e_{ij}|\phi_{ij})].$$

Unlike truncations of the global stick-breaking process [4], our variational bounds are *nested*, so that lower-order approximations are special cases of higher-order ones with some zeroed parameters.

## 3.2  Structured Variational Inference with Linear Time and Storage Complexity

Conventional, coordinate ascent variational inference algorithms iteratively optimize each parameter given fixed values for all others. Community membership and interaction parameters are updated as

$$\lambda_{ka} = \tau_a + \sum_{ij}^E \sum_{k=1}^K \phi_{ijkk} y_{ij}, \qquad \lambda_{kb} = \tau_b + \sum_{ij}^E \sum_{k=1}^K \phi_{ijkk}(1 - y_{ij}), \tag{6}$$

$$\theta_{ik} = \alpha\beta_k + \sum_{(i,j)\in E} \sum_{\ell=1}^K \phi_{ijk\ell}. \tag{7}$$

Here, the final summation is over all potential edges $(i, j)$ linked to node $i$. Updates for assignment distributions depend on expectations of log community assignment probabilities:

$$\mathbb{E}_q[\log(w_k)] = \psi(\lambda_{ka}) - \psi(\lambda_{ka} + \lambda_{kb}), \qquad \mathbb{E}_q[\log(1 - w_k)] = \psi(\lambda_{kb}) - \psi(\lambda_{ka} + \lambda_{kb}), \quad (8)$$

$$\tilde{\pi}_{ik} \triangleq \exp\{\mathbb{E}_q[\log(\pi_{ik})]\} = \exp\{\psi(\theta_{ik}) - \psi(\sum_{\ell=1}^{K+1} \theta_{i\ell})\}, \qquad \tilde{\pi}_i \triangleq \sum_{k=1}^{K} \tilde{\pi}_{ik}. \quad (9)$$

Given these sufficient statistics, the assignment distributions can be updated as follows:

$$\phi_{ijkk} \propto \tilde{\pi}_{ik} \tilde{\pi}_{jk} f(w_k, y_{ij}), \qquad (10)$$

$$\phi_{ijk\ell} \propto \tilde{\pi}_{ik} \tilde{\pi}_{j\ell} f(\epsilon, y_{ij}), \quad \ell \neq k. \qquad (11)$$

Here, $f(w_k, y_{ij}) = \exp\{y_{ij}\mathbb{E}_q[\log(w_k)] + (1 - y_{ij})\mathbb{E}_q[\log(1 - w_k)]\}$. More detailed derivations of related updates have been developed for the MMSB [1].

A naive implementation of these updates would require $\mathcal{O}(K^2)$ computation and storage for each assignment distribution $q(e_{ij} \mid \phi_{ij})$. Note, however, that the updates for $q(w_k \mid \lambda_k)$ in Eq. (6) depend only on the $K$ probabilities $\phi_{ijkk}$ that nodes select the same community. Using the updates for $\phi_{ijk\ell}$ from Eq. (11), the update of $q(\pi_i \mid \theta_i)$ in Eq. (7) can be expanded as follows:

$$\theta_{ik} = \alpha\beta_k + \sum_{(i,j)\in E} \phi_{ijkk} + \frac{1}{Z_{ij}} \sum_{\ell \neq k} \tilde{\pi}_{ik} \tilde{\pi}_{j\ell} f(\epsilon, y_{ij}) \qquad (12)$$

$$= \alpha\beta_k + \sum_{(i,j)\in E} \phi_{ijkk} + \frac{1}{Z_{ij}} \tilde{\pi}_{ik} f(\epsilon, y_{ij})(\tilde{\pi}_j - \tilde{\pi}_{jk}).$$

Note that $\tilde{\pi}_j$ need only be computed once, in $\mathcal{O}(K)$ operations. The normalization constant $Z_{ij}$, which is defined so that $\phi_{ij}$ is a valid categorical distribution, can also be computed in linear time:

$$Z_{ij} = \tilde{\pi}_i \tilde{\pi}_j f(\epsilon, y_{ij}) + \sum_{k=1}^{K} \tilde{\pi}_{ik} \tilde{\pi}_{jk}(f(w_k, y_{ij}) - f(\epsilon, y_{ij})). \qquad (13)$$

Finally, to evaluate our variational bound and assess algorithm convergence, we still need to calculate the likelihood and entropy terms dependent on $\phi_{ijk\ell}$. However, we can compute part of our bound by caching our partition function $Z_{ij}$ in linear time. See $\ddagger A.2$ for details regarding the full derivation of this ELBO and its extensions.

### 3.3 Stochastic Variational Inference

Standard variational batch updates become computationally intractable when $N$ becomes very large. Recent advancements in applying stochastic optimization techniques within variational inference [8] showed that if our variational mean-field family of distributions are members of the exponential family, we can derive a simple stochastic natural gradient update for our global parameters $\lambda, \theta, v$. These gradients can be calculated from only a subset of the data and are noisy approximations of the true natural gradient for the variational objective, but represent an unbiased estimate of that gradient.

To accomplish this, we define a new variational objective with respect to our current set of observations. This function, in expectation, is equivalent to our true ELBO. By taking natural gradients with respect to our new variational objective for our global variables $\lambda, \theta$, we have

$$\nabla\lambda_{ka}^* = \frac{1}{g(i,j)} \phi_{ijkk} y_{ij} + \tau_a - \lambda_{ka}; \qquad (14)$$

$$\nabla\theta_{ik}^* = \frac{1}{g(i,j)} \sum_{(i,j)\in E} \sum_{\ell=1}^{K} \phi_{ijk\ell} + \alpha\beta_k - \theta_{ik}, \qquad (15)$$

where the natural gradient for $\nabla\lambda_{kb}^*$ is symmetric to $\nabla\lambda_{ka}^*$ and where $y_{ij}$ in Eq. (14) is replaced by $(1 - y_{ij})$. Note that $\sum_{(i,j)\in E} \sum_{\ell=1}^{K} \phi_{ijk\ell}$ was shown in the previous section to be computable in $\mathcal{O}(K)$. The scaling term $g(i, j)$ is needed for an unbiased update to our expectation. If $g(i, j) = 2/N(N - 1)$, then this would represent a uniform distribution over possible edge selections in our undirected graphs. In general, $g(i, j)$ can be an arbitrary distribution over possible edge selections such as a distribution over sets of edges as long as the expectation with respect to this distribution is equivalent to the original ELBO [6]. When referring to the scaling constant associated with sets, we consider the notation of $h(T)$ instead of $g(i, j)$.

We optimize this ELBO with a Robbins-Monro algorithm which iteratively steps along the direction of this noisy gradient. We specify a learning rate $\rho_t \triangleq (\mu_0 + t)^{-\kappa}$ at time $t$ where $\kappa \in (.5, 1]$ and $\mu_0 \geq 0$ downweights the influence of earlier updates. With the requirement that $\sum_t \rho_t^2 < \infty$ and

$\sum_t \rho_t = \infty$, we will provably converge to a local optimum. For our global variational parameters $\{\lambda, \theta\}$, the updates at iteration $t$ are now

$$\lambda_{ka}^t = \lambda_{ka}^{t-1} + \rho_t(\nabla \lambda_{ka}^*) = (1 - \rho_t)\lambda_{ka}^{t-1} + \rho_t(\frac{1}{g(i,j)}\phi_{ijkk}y_{ij} + \tau_a); \qquad (16)$$

$$\theta_{ik}^t = \theta_{ik}^{t-1} + \rho_t(\nabla \theta_{ik}^*) = (1 - \rho_t)\theta_{ik}^{t-1} + \rho_t(\frac{1}{g(i,j)}\sum_{(i,j)\in E}\sum_{\ell=1}^K \phi_{ijk\ell} + \alpha\beta_k); \qquad (17)$$

$$v_k^t = (1 - \rho_t)v_k^{t-1} + \rho_t(v_k^*), \qquad (18)$$

where $v_k^*$ is obtained via a constrained optimization task using the gradients derived in $\ddagger A.3$. Defining an update on our global parameters given a single edge observation can result in very poor local optima. In practice, we specify a mini-batch $T$, a set of unique observations in determining a noisy gradient that is more informative. This results in a simple summation over the sufficient statistics associated with the set of observations as well as a change to $g(i,j)$ to reflect the necessary scaling of our gradients when we can no longer assume our samples are uniformly chosen from our dataset.

### 3.4 Restricted Stratified Node Sampling

Stochastic variational inference provides us with the ability to choose a sampling scheme that allows us to better exploit the sparsity of real world networks. Given the success of stratified node sampling [6], we consider this technique for all our experiments. Briefly, stratified node-sampling randomly selects a single node $i$ and either chooses its associated links or a set of edges from $m$ equally sized non-link edge sets. For this mini-batch strategy, $h(T) = 1/N$ for link sets and $h(T) = 1/Nm$ for a partitioned non-link set. In [6], all nodes in $\pi$ were considered global parameters and updated after each mini-batch. For our model, we also treat $\pi$ similarly, but maintain a separate learning rate $\rho_i$ for each node. This allows us to focus on updating only nodes that are relevant to our mini-batch as well as limit the computational costs associated with this global update. To ensure that our Robbins-Monro conditions are still satisfied, we set the learning rate for nodes that are not part of our mini-batch to be 0. When a new minibatch contains this particular node, we look to the most previous learning rate and assume this value as the previous learning rate. This modified subsequence of learning rates maintains our convergence criterion so that the $\sum_t \rho_{it}^2 < \infty$ and that $\sum_t \rho_{it} = \infty$. We show how performing this simple modification results in significant improvements in both perplexity and link prediction scores.

### 3.5 Pruning Moves

Our nested truncation requires setting an initial number of communities $K$. A large truncation lets the posterior find the best number of communities, but can be computationally costly. A truncation set too small may not be expressive enough to capture the best approximate posterior. To remedy this, we define a set of pruning moves aimed at improving inference by removing communities that have very small posterior mass. Pruning moves provide the model with a more parsimonious and interpretable latent structure, and may also significantly reduce the computational cost of subsequent iterations. Figure 2 provides an example illustrating how pruning occurs in our model.

To determine communities which are good candidates for pruning, for each community $k$ we first compute $\Theta_k = (\sum_{i=1}^N \theta_{ik})/(\sum_{i=1}^N \sum_{k=1}^K \theta_{ik})$. Any community for which $\Theta_k < (\log K)/N$ for $t^* = N/2$ consecutive iterations is then evaluated in more depth. We scale $t^*$ with the number of nodes $N$ within the graph to ensure that a broad set of observations are accounted for. To estimate an approximate but still informative ELBO for the pruned model, we must associate a set of relevant observations to each pruning candidate. In particular, we approximate the pruned ELBO $\mathcal{L}(q^{\text{prune}})$ by considering observations $y_{ij}$ among pairs of nodes with significant mass in the pruned community. We also calculate $\mathcal{L}(q^{old})$ from these same observations, but with the old model parameters. We then compare these two values to accept or reject the pruning of the low-weight community.

## 4 Experiments

In this section we perform experiments that compare the performance of the aHDPR model to the aMMSB. We show significant gains in AUC and perplexity scores by using the restricted form of

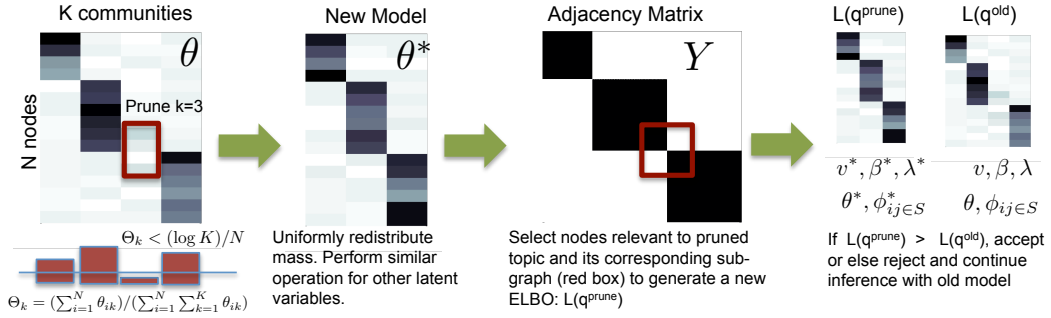

K communities

$\theta$

Prune k=3

N nodes

$\Theta_k < (\log K)/N$

$\Theta_k = (\sum_{i=1}^{N} \theta_{ik})/(\sum_{i=1}^{N} \sum_{k=1}^{K} \theta_{ik})$

New Model

$\theta^*$

Uniformly redistribute mass. Perform similar operation for other latent variables.

Adjacency Matrix

$Y$

Select nodes relevant to pruned topic and its corresponding sub-graph (red box) to generate a new ELBO: L(q$^{\text{prune}}$).

L(q$^{\text{prune}}$)    L(q$^{\text{old}}$)

$v^*, \beta^*, \lambda^*$     $v, \beta, \lambda$
$\theta^*, \phi^*_{ij \in S}$     $\theta, \phi_{ij \in S}$

If L(q$^{\text{prune}}$) > L(q$^{\text{old}}$), accept or else reject and continue inference with old model

Figure 2: Pruning extraneous communities. Suppose that community $k = 3$ is considered for removal. We specify a new model by redistributing its mass $\sum_{i=1}^{N} \theta_{i3}$ uniformly across the remaining communities $\theta_{i\ell}, \ell \neq 3$. An analogous operation is used to generate $\{v^*, \beta^*, \lambda^*_a, \lambda^*_b, \theta^*\}$. To accurately estimate the true change in ELBO for this pruning, we select the $n^* = 10$ nodes with greatest participation $\theta_{i3}$ in community 3. Let $S$ denote the set of all pairs of these nodes, and $y_{ij \in S}$ their observed relationships. From these observations we can estimate $\phi^*_{ij \in S}$ for a model in which community $k = 3$ is pruned, and a corresponding ELBO $\mathcal{L}(q^{\text{prune}})$. Using the data from the same sub-graph, but the old un-pruned model parameters, we estimate an alternative ELBO $\mathcal{L}(q^{old})$. We accept if $\mathcal{L}(q^{\text{prune}}) > \mathcal{L}(q^{old})$, and reject otherwise. Because our structured mean-field approach provides simple direct updates for $\phi^*_{ij \in S}$, the calculation of $\mathcal{L}(q^{old})$ and $\mathcal{L}(q^{\text{prune}})$ is efficient.

stratified node sampling, a quick K-means initialization[1] for $\theta$, and our efficient structured mean-field approach combined with pruning moves. We perform a detailed comparison on a synthetic toy dataset, as well as the real-world relativity collaboration network, using a variety of metrics to show the benefits of each contribution. We then show significant improvements over the baseline aMMSB model in both AUC and perplexity metrics on several real-world datasets previously analyzed by [6]. Finally, we perform a qualitative analysis on the LittleSis network and demonstrate the usefulness of using our learned latent community structure to create visualizations of large networks. For additional details on the parameters used in these experiments, please see ‡$A.1$.

## 4.1 Synthetic and Collaboration Networks

The synthetic network we use for testing is generated from the standards and software outlined in [12] to produce realistic synthetic networks with overlapping communities and power-law degree distributions. For these purposes, we set the number of nodes $N = 1000$, with the minimum degree per node set to 10 and its maximum to 60. On this network the true number of latent communities was found to be $K = 56$. Our real world networks include 5 undirected networks originally ranging from $N = 5,242$ to $N = 27,770$. These raw networks, however, contain several disconnected components. Both the aMMSB and aHDPR achieve highest posterior probability by assigning each connected component distinct, non-overlapping communities; effectively, they analyze each connected sub-graph independently. To focus on the more challenging problem of identifying overlapping community structure, we take the largest connected component of each graph for analysis.

**Initialization and Node-Specific Learning Rates.** The upper-left panels in Fig. 3 compare different aHDPR inference algorithms, and the perplexity scores achieved on various networks. Here we demonstrate the benefits of initializing $\theta$ via K-means, and our restricted stratified node sampling procedure. For our random initializations, we initalized $\theta$ in the same fashion as the aMMSB. Using a combination of both modifications, we achieve the best perplexity scores on these datasets. The node-specific learning rates intuitively restrict updates for $\theta$ to batches containing relevant observations, while our K-means initialization quickly provides a reasonable single-membership partition as a starting point for inference.

**Naive Mean-Field vs. Structured Mean-Field.** The naive mean-field approach is the aHDPR model where the community indicator assignments are split into $s_{ij}$ and $r_{ij}$. This can result in severe local optima due to their coupling as seen in some experiments in Fig. 4. The aMMSB in some

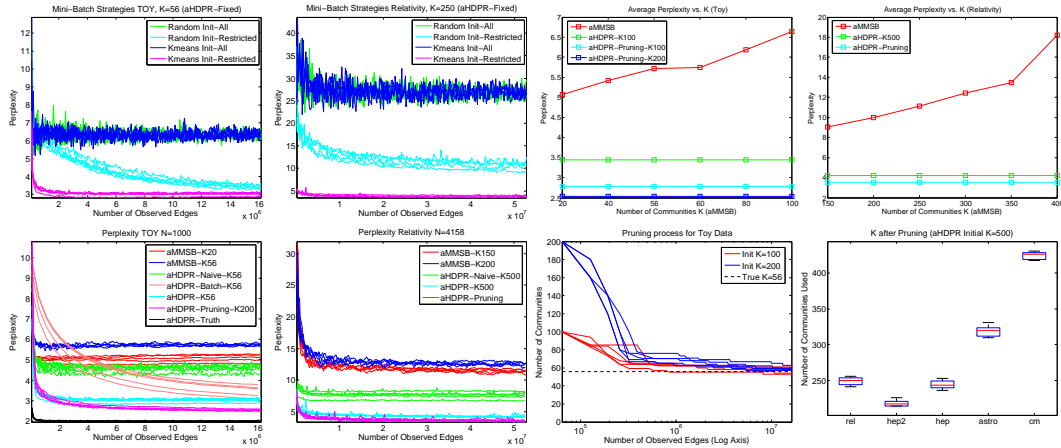

Figure 3: The upper left shows benefits of a restricted update and a K-means initialization for stratified node sampling on both synthetic and relativity networks. The upper right shows the sensitivity of the aMMSB as $K$ varies versus the aHDPR. The lower left shows various perplexity scores for the synthetic and relativity networks with the best performing model (aHDPR-Pruning) scoring an average AUC of $0.9675 \pm .0017$ on the synthetic network and $0.9466 \pm .0062$ on the relativity network. The lower right shows the pruning process for the toy data and the final $K$ communities discovered on our real-world networks.

instances performs better than the naive mean-field approach, but this can be due to differences in our initialization procedures. However, by changing our inference procedure to an efficient structured mean-field approach, we see significant improvements across all datasets.

**Benefits of Pruning Moves.** Pruning moves were applied every $N/2$ iterations with a maximum of $K/10$ communities removed per move. If the number of prune candidates was greater than $K/10$, then $K/10$ communities with the lowest mass were chosen. The lower right portion of Fig. 3 shows that our pruning moves can learn close to the true underlying number of clusters (K=56) on a synthetic network even when significantly altering its initial $K$. Across several real world networks, there was low variance between runs with respect to the final $K$ communities discovered, suggesting a degree of robustness. Furthermore, pruning moves improved perplexity and AUC scores across every dataset as well as reducing computational costs during inference.

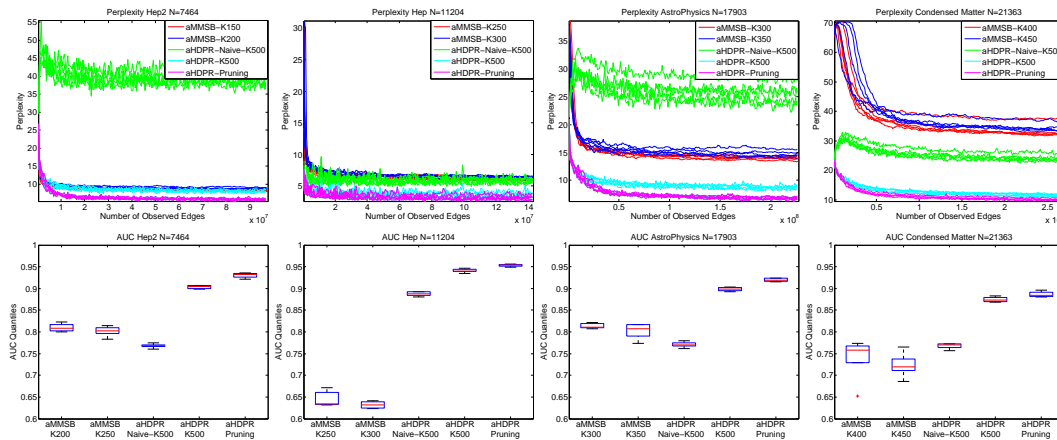

Figure 4: Analysis of four real-world collaboration networks. The figures above show that the aHDPR with pruning moves has the best performance, in terms of both perplexity (top) and AUC (bottom) scores.

## 4.2 The LittleSis Network

The LittleSis network was extracted from the website (http://littlesis.org), which is an organization that acts as a watchdog network to connect the dots between the world's most powerful people

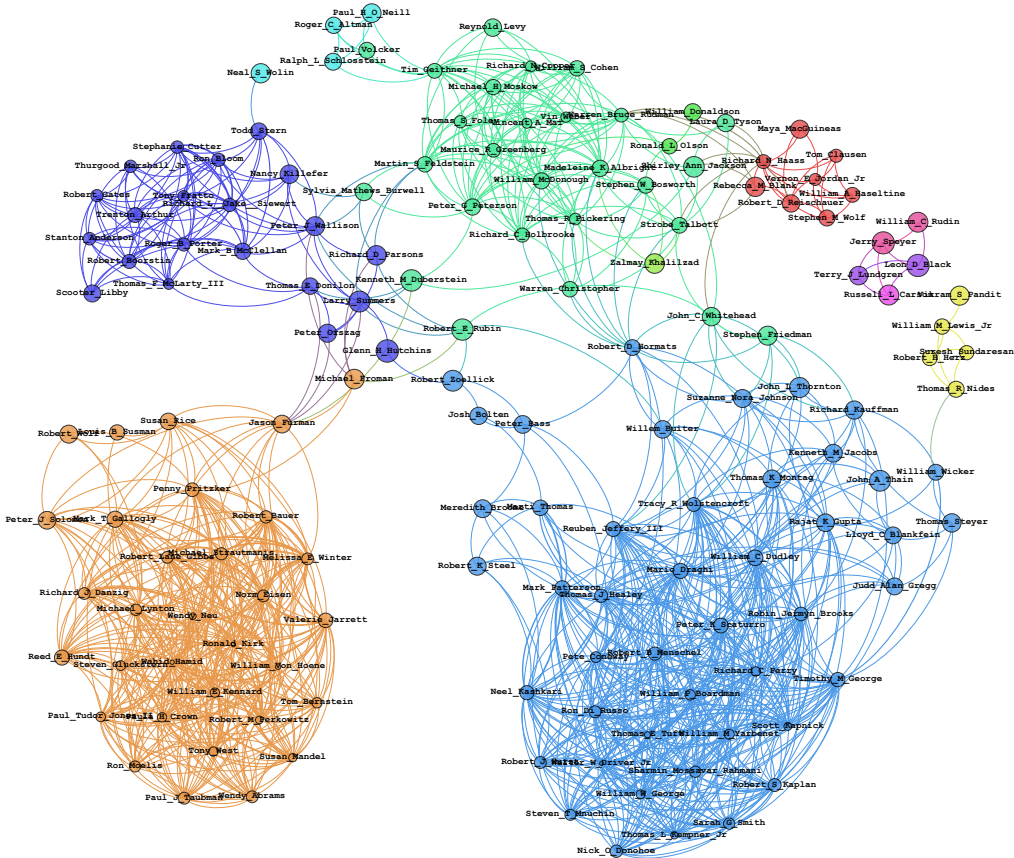

Figure 5: **The LittleSis Network**. Near the center in violet we have prominent government figures such as Larry H. Summers (71st US Treasury Secretary) and Robert E. Rubin (70th US Treasury Secretary) with ties to several distinct communities, representative of their high posterior bridgness. Conversely, within the beige colored community, individuals with small posterior bridgness such as Wendy Neu can reflect a career that was highly focused in one organization. A quick internet search shows that she is currently the CEO of Hugo Neu, a green-technology firm where she has worked for over 30 years. An analysis on this type of network might provide insights into the structures of power that shape our world and the key individuals that define them.

and organizations. Our final graph contained 18,831 nodes and 626,881 edges, which represents a relatively sparse graph with edge density of $0.35\%$ (for details on how this dataset was processed see ‡$A.3$). For this analysis, we ran the aHDPR with pruning on the entire dataset using the same settings from our previous experiments. We then took the top 200 degree nodes and generated weighted edges based off of a variational distance between their learned expected variational posteriors such that $d_{ij} = 1 - \frac{|\mathbb{E}_q[\pi_i] - \mathbb{E}_q[\pi_j]|}{2}$. This weighted edge was then included in our visualization software [3] if $d_{ij} > 0.5$. Node sizes were determined by posterior bridgness [16] where $b_i = 1 - \sqrt{K/(K-1)} \sum_{k=1}^{K} (\mathbb{E}_q[\pi_{ik}] - \frac{1}{K})^2$ and measures the extent to which a node is involved with multiple communities. Larger nodes have greater posterior bridgeness while node colors represent its dominant community membership. Our learned latent communities can drive these types of visualizations that otherwise might not have been possible given the raw subgraph (see ‡$A.4$).

## 5 Discussion

Our model represents the first Bayesian nonparametric relational model to use a stochastic variational approach for efficient inference. Our pruning moves allow us to save computation and improve inference in a principled manner while our efficient structured mean-field inference procedure helps us escape local optima. Future extensions of interest could entail advanced split-merge moves that can grow the number of communities as well as extending these scalable inference algorithms to more sophisticated relational models.

## Footnotes

[1]Our K-means initialization views the rows of the adjacency matrix as distinct data points and produces a single community assignment $z_i$ for each node. To initialize community membership distributions based on these assignments, we set $\theta_{iz_i} = N - 1$ and $\theta_{i \setminus z_i} = \alpha$.

# References

[1] E. Airoldi, D. Blei, S. Fienberg, and E. Xing. Mixed membership stochastic blockmodels. *JMLR*, 9, 2008.

[2] S. Amari. Natural gradient works efficiently in learning. *Neural Computation*, 10(2):251–276, 1998.

[3] M. Bastian, S. Heymann, and M. Jacomy. Gephi: An open source software for exploring and manipulating networks, 2009.

[4] D. M. Blei and M. I. Jordan. Variational methods for the dirichlet process. In *ICML*, 2004.

[5] M. Bryant and E. B. Sudderth. Truly nonparametric online variational inference for hierarchical dirichlet processes. In *NIPS*, pages 2708–2716, 2012.

[6] P. Gopalan, D. M. Mimno, S. Gerrish, M. J. Freedman, and D. M. Blei. Scalable inference of overlapping communities. In *NIPS*, pages 2258–2266, 2012.

[7] T. L. Griffiths and Z. Ghahramani. Infinite latent feature models and the Indian buffet process. Technical Report 2005-001, Gatsby Computational Neuroscience Unit, May 2005.

[8] M. Hoffman, D. Blei, C. Wang, and J. Paisley. Stochastic variational inference. *arXiv preprint arXiv:1206.7051*, 2012.

[9] M. D. Hoffman, D. M. Blei, and F. R. Bach. Online learning for latent dirichlet allocation. In *NIPS*, pages 856–864, 2010.

[10] C. Kemp, J. Tenenbaum, T. Griffiths, T. Yamada, and N. Ueda. Learning systems of concepts with an infinite relational model. In *AAAI*, 2006.

[11] D. Kim, M. C. Hughes, and E. B. Sudderth. The nonparametric metadata dependent relational model. In *ICML*, 2012.

[12] A. Lancichinetti and S. Fortunato. Benchmarks for testing community detection algorithms on directed and weighted graphs with overlapping communities. *Phys. Rev. E*, 80(1):016118, July 2009.

[13] littlesis.org. Littlesis is a free database detailing the connections between powerful people and organizations, June 2009.

[14] K. Miller, T. Griffiths, and M. Jordan. Nonparametric latent feature models for link prediction. In *NIPS*, 2009.

[15] M. Morup, M. N. Schmidt, and L. K. Hansen. Infinite multiple membership relational modeling for complex networks. In *Machine Learning for Signal Processing (MLSP), 2011 IEEE International Workshop on*, pages 1–6. IEEE, 2011.

[16] T. Nepusz, A. Petrczi, L. Ngyessy, and F. Bazs. Fuzzy communities and the concept of bridgeness in complex networks. *Phys Rev E Stat Nonlin Soft Matter Phys*, 77(1 Pt 2):016107, 2008.

[17] M. Sato. Online model selection based on the variational bayes. *Neural Computation*, 13(7):1649–1681, 2001.

[18] Y. W. Teh, M. I. Jordan, M. J. Beal, and D. M. Blei. Hierarchical Dirichlet processes. *JASA*, 101(476):1566–1581, Dec. 2006.

[19] Y. W. Teh, K. Kurihara, and M. Welling. Collapsed variational inference for hdp. In *NIPS*, 2007.

[20] Y. Wang and G. Wong. Stochastic blockmodels for directed graphs. *JASA*, 82(397):8–19, 1987.

